# Co-regularized Multi-view Spectral Clustering

**Abhishek Kumar**[*]
Dept. of Computer Science
University of Maryland,
College Park, MD
abhishek@cs.umd.edu

**Piyush Rai**[*]
Dept. of Computer Science
University of Utah,
Salt Lake City, UT
piyush@cs.utah.edu

**Hal Daumé III**
Dept. of Computer Science
University of Maryland,
College Park, MD
hal@umiacs.umd.edu

## Abstract

In many clustering problems, we have access to multiple views of the data each of which could be individually used for clustering. Exploiting information from multiple views, one can hope to find a clustering that is more accurate than the ones obtained using the individual views. Often these different views admit same underlying clustering of the data, so we can approach this problem by looking for clusterings that are consistent across the views, i.e., corresponding data points in each view should have same cluster membership. We propose a spectral clustering framework that achieves this goal by co-regularizing the clustering hypotheses, and propose two co-regularization schemes to accomplish this. Experimental comparisons with a number of baselines on two synthetic and three real-world datasets establish the efficacy of our proposed approaches.

## 1 Introduction

Many real-world datasets have representations in the form of multiple views [1, 2]. For example, webpages usually consist of both the page-text and hyperlink information; images on the web have captions associated with them; in multi-lingual information retrieval, the same document has multiple representations in different languages, and so on. Although these individual *views* might be sufficient on their own for a given learning task, they can often provide complementary information to each other which can lead to improved performance on the learning task at hand.

In the context of data clustering, we seek a partition of the data based on some similarity measure between the examples. Our of the numerous clustering algorithms, Spectral Clustering has gained considerable attention in the recent past due to its strong performance on arbitrary shaped clusters, and due to its well-defined mathematical framework [3]. Spectral clustering is accomplished by constructing a graph from the data points with edges between them representing the similarities, and solving a relaxation of the normalized min-cut problem on this graph [4]. For the multi-view clustering problem, we work with the assumption that the true underlying clustering would assign corresponding points in each view to the same cluster. Given this assumption, we can approach the multi-view clustering problem by limiting our search to clusterings that are compatible across the graphs defined over each of the views: corresponding nodes in each graph should have the same cluster membership.

In this paper, we propose two spectral clustering algorithms that achieve this goal by *co-regularizing* the clustering hypotheses across views. Co-regularization is a well-known technique in semi-supervised literature; however, not much is known on using it for unsupervised learning problems. We propose novel spectral clustering objective functions that *implicitly* combine graphs from multiple views of the data to achieve a better clustering. Our proposed methods give us a way to combine multiple kernels (or similarity matrices) for the clustering problem. Moreover, we would like to note here that although multiple kernel learning has met with considerable success on supervised learning problems, similar investigations for unsupervised learning have been found lacking so far, which is one of the motivations behind this work.

---

[*]Authors contributed equally

## 2 Co-regularized Spectral Clustering

We assume that we are given data having multiple representations (i.e., views). Let $\mathbf{X} = \{\mathbf{x}_1^{(v)}, \mathbf{x}_2^{(v)}, \ldots, \mathbf{x}_n^{(v)}\}$ denote the examples in view $v$ and $\mathbf{K}^{(v)}$ denote the similarity or kernel matrix of $\mathbf{X}$ in this view. We write the normalized graph Laplacian for this view as: $\mathcal{L}^{(v)} = \mathbf{D}^{(v)^{-1/2}} \mathbf{K}^{(v)} \mathbf{D}^{(v)^{-1/2}}$. The *single view* spectral clustering algorithm of [5] solves the following optimization problem for the normalized graph Laplacian $\mathbf{L}^{(v)}$:

$$\max_{\mathbf{U}^{(v)} \in \mathbb{R}^{n \times k}} tr\left(\mathbf{U}^{(v)^T} \mathcal{L}^{(v)} \mathbf{U}^{(v)}\right), \quad \text{s.t.} \quad \mathbf{U}^{(v)^T} \mathbf{U}^{(v)} = I \tag{1}$$

where $tr$ denotes the matrix trace. The rows of matrix $\mathbf{U}^{(v)}$ are the embeddings of the data points that can be given to the $k$-means algorithm to obtain cluster memberships. For a detailed introduction to both theoretical and practical aspects of spectral clustering, the reader is referred to [3]. Our multi-view spectral clustering framework builds on the standard spectral clustering with a single view, by appealing to the co-regularization framework typically used in the semi-supervised learning literature [1].

Co-regularization in semi-supervised learning essentially works by making the hypotheses learned from different views of the data agree with each other on unlabeled data [6]. The framework employs two main assumptions for its success: (a) the true target functions in each view should agree on the labels for the unlabeled data *(compatibility)*, and (b) the views are independent given the class label *(conditional independence)*. The *compatibility* assumption allows us to shrink the space of possible target hypotheses by searching only over the compatible functions. Standard PAC-style analysis [1] shows that this also leads to reductions in the number of examples needed to learn the target function, since this number depends on the size of the hypothesis class. The *independence* assumption makes it unlikely for compatible classifiers to agree on wrong labels. In the case of clustering, this would mean that a data point in both views would be assigned to the correct cluster with high probability.

Here, we propose two co-regularization based approaches to make the clustering hypotheses on different graphs (i.e., views) agree with each other. The effectiveness of spectral clustering hinges crucially on the construction of the graph Laplacian and the resulting eigenvectors that reflect the cluster structure in the data. Therefore, we construct an objective function that consists of the graph Laplacians from all the views of the data and regularize on the eigenvectors of the Laplacians such that the cluster structures resulting from each Laplacian look consistent across all the views.

Our first co-regularization scheme (Section 2.1) enforces that the eigenvectors $\mathbf{U}^{(v)}$ and $\mathbf{U}^{(w)}$ of a view pair $(v, w)$ should have high pairwise similarity (using a pair-wise co-regularization criteria we will define in Section 2.1). Our second co-regularization scheme (Section 2.3) enforces the view-specific eigenvectors to look similar by regularizing them towards a common *consensus* (centroid based co-regularization). The idea is different from previously proposed consensus clustering approaches [7] that commit to individual clusterings in the first step and then combine them to a consensus in the second step. We optimize for individual clusterings as well as the consensus using a joint cost function.

### 2.1 Pairwise Co-regularization

In standard spectral clustering, the eigenvector matrix $\mathbf{U}^{(v)}$ is the data representation for subsequent $k$-means clustering step (with $i$'th row mapping to the original $i$'th sample). In our proposed objective function, we encourage the pairwise similarities of examples under the new representation (in terms of rows of $\mathbf{U}^{(\cdot)}$'s) to be similar across all the views. This amounts to enforcing the spectral clustering hypotheses (which are based on the $\mathbf{U}^{(\cdot)}$'s) to be the same across all the views.

We will work with two-view case for the ease of exposition. This will later be extended to more than two views. We propose the following cost function as a measure of disagreement between clusterings of two views:

$$D(\mathbf{U}^{(v)}, \mathbf{U}^{(w)}) = \left\| \frac{\mathbf{K}_{\mathbf{U}^{(v)}}}{||\mathbf{K}_{\mathbf{U}^{(v)}}||_F^2} - \frac{\mathbf{K}_{\mathbf{U}^{(w)}}}{||\mathbf{K}_{\mathbf{U}^{(w)}}||_F^2} \right\|_F^2. \tag{2}$$

$\mathbf{K}_{\mathbf{U}^{(v)}}$ is the similarity matrix for $\mathbf{U}^{(v)}$, and $|| \cdot ||_F$ denotes the Frobenius norm of the matrix. The similarity matrices are normalized by their Frobenius norms to make them comparable across

views. We choose linear kernel, i.e., $k(\mathbf{x}_i, \mathbf{x}_j) = \mathbf{x}_i^T \mathbf{x}_j$ as our similarity measure in Equation 2. This implies that we have $\mathbf{K}_{\mathbf{U}^{(v)}} = \mathbf{U}^{(v)}\mathbf{U}^{(v)^T}$. The reason for choosing linear kernel to measure similarity of $\mathbf{U}^{(\cdot)}$ is twofold. First, the similarity measure (or kernel) used in the Laplacian for spectral clustering has already taken care of the non-linearities present in the data (if any), and the embedding $\mathbf{U}^{(\cdot)}$ being real-valued cluster indicators, can be considered to obey linear similarities. Secondly, we get a nice optimization problem by using linear kernel for $\mathbf{U}^{(\cdot)}$. We also note that $||\mathbf{K}_{\mathbf{U}^{(v)}}||_F^2 = k$, where $k$ is the number of clusters. Substituting this in Equation 2 and ignoring the constant additive and scaling terms that depend on the number of clusters, we get

$$D(\mathbf{U}^{(v)}, \mathbf{U}^{(w)}) = -tr\left(\mathbf{U}^{(v)}\mathbf{U}^{(v)^T}\mathbf{U}^{(w)}\mathbf{U}^{(w)^T}\right)$$

We want to minimize the above disagreement between the clusterings of views $v$ and $w$. Combining this with the spectral clustering objectives of individual views, we get the following joint *maximization* problem for two graphs:

$$\max_{\substack{\mathbf{U}^{(v)} \in \mathbb{R}^{n \times k} \\ \mathbf{U}^{(w)} \in \mathbb{R}^{n \times k}}} tr\left(\mathbf{U}^{(v)^T}\mathcal{L}^{(v)}\mathbf{U}^{(v)}\right) + tr\left(\mathbf{U}^{(w)^T}\mathcal{L}^{(w)}\mathbf{U}^{(w)}\right) + \lambda\, tr\left(\mathbf{U}^{(v)}\mathbf{U}^{(v)^T}\mathbf{U}^{(w)}\mathbf{U}^{(w)^T}\right)$$

$$\text{s.t.}\quad \mathbf{U}^{(v)^T}\mathbf{U}^{(v)} = I,\ \mathbf{U}^{(w)^T}\mathbf{U}^{(w)} = I \tag{3}$$

The hyperparameter $\lambda$ trades-off the spectral clustering objectives and the spectral embedding (dis)agreement term. The joint optimization problem given by Equation 3 can be solved using alternating maximization w.r.t. $\mathbf{U}^{(v)}$ and $\mathbf{U}^{(w)}$. For a given $\mathbf{U}^{(w)}$, we get the following optimization problem in $\mathbf{U}^{(v)}$:

$$\max_{\mathbf{U}^{(v)} \in \mathbb{R}^{n \times k}} tr\left\{\mathbf{U}^{(v)^T}\left(\mathcal{L}^{(v)} + \lambda\mathbf{U}^{(w)}\mathbf{U}^{(w)^T}\right)\mathbf{U}^{(v)}\right\},\quad \text{s.t.}\quad \mathbf{U}^{(v)^T}\mathbf{U}^{(v)} = I. \tag{4}$$

This is a standard spectral clustering objective on view $v$ with graph Laplacian $\mathcal{L}^{(v)} + \lambda\mathbf{U}^{(w)}\mathbf{U}^{(w)^T}$. This can be seen as a way of combining kernels or Laplacians. The difference from standard kernel combination (kernel addition, for example) is that the combination is adaptive since $\mathbf{U}^{(w)}$ keeps getting updated at each step, as guided by the clustering algorithm. The solution $\mathbf{U}^{(v)}$ is given by the top-$k$ eigenvectors of this modified Laplacian. Since the alternating maximization can make the algorithm stuck in a local maximum [8], it is important to have a sensible initialization. If there is no prior information on which view is more *informative* about the clustering, we can start with any of the views. However, if we have some a priori knowledge on this, we can start with the graph Laplacian $\mathcal{L}^{(w)}$ of the more informative view and initialize $\mathbf{U}^{(w)}$. The alternating maximization is carried out after this until convergence. Note that one possibility could be to regularize directly on the eigenvectors $\mathbf{U}^{(v)}$'s and make them close to each other (e.g., in the sense of the Frobenious norm of the difference between $\mathbf{U}^{(v)}$ and $\mathbf{U}^{(w)}$). However, this type of regularization could be too restrictive and could end up shrinking the hypothesis space of feasible clusterings too much, thus ruling out many valid clusterings.

For fixed $\lambda$ and $n$, the joint objective of Eq. 3 can be shown to be bounded from above by a constant. Since the objective is non-decreasing with the iterations, the algorithm is guaranteed to converge. In practice, we monitor the convergence by the difference in the value of the objective between consecutive iterations, and stop when the difference falls below a minimum threshold of $\epsilon = 10^{-4}$. In all our experiments, we converge within less than 10 iterations. Note that we can use either $\mathbf{U}^{(v)}$ or $\mathbf{U}^{(w)}$ in the final $k$-means step of the spectral clustering algorithm. In our experiments, we note a marginal difference in the clustering performance depending on which $\mathbf{U}^{(\cdot)}$ is used in the final step of $k$-means clustering.

## 2.2 Extension to Multiple Views

We can extend the co-regularized spectral clustering proposed in the previous section for more than two views. This can be done by employing pair-wise co-regularizers in the objective function of Eq. 3. For $m$ number of views, we have

$$\max_{\mathbf{U}^{(1)}, \mathbf{U}^{(2)}, \ldots, \mathbf{U}^{(m)} \in \mathbb{R}^{n \times k}} \sum_{v=1}^{m} tr\left(\mathbf{U}^{(v)^T}\mathcal{L}^{(v)}\mathbf{U}^{(v)}\right) + \lambda \sum_{\substack{1 \le v, w \le m \\ v \ne w}} tr\left(\mathbf{U}^{(v)}\mathbf{U}^{(v)^T}\mathbf{U}^{(w)}\mathbf{U}^{(w)^T}\right),$$

$$\text{s.t.}\quad \mathbf{U}^{(v)^T}\mathbf{U}^{(v)} = I,\ \forall\, 1 \le v \le V \tag{5}$$

We use a common $\lambda$ for all pair-wise co-regularizers for simplicity of exposition, however different $\lambda$'s can be used for different pairs of views. Similar to the two-view case, we can optimize it by alternating maximization cycling over the views. With all but one $\mathbf{U}^{(v)}$ fixed, we have the following optimization problem:

$$\max_{\mathbf{U}^{(v)}} tr \left\{ \mathbf{U}^{(v)^T} \left( \mathcal{L}^{(v)} + \lambda \sum_{\substack{1 \leq w \leq m, \\ w \neq v}} \mathbf{U}^{(w)} \mathbf{U}^{(w)^T} \right) \mathbf{U}^{(v)} \right\}, \quad \text{s.t.} \quad \mathbf{U}^{(v)^T} \mathbf{U}^{(v)} = I \quad (6)$$

We initialize all $\mathbf{U}^{(v)}$, $2 \leq v \leq m$ by solving the spectral clustering problem for single views. We solve the objective of Eq. 6 for $\mathbf{U}^{(1)}$ given all other $\mathbf{U}^{(v)}$, $2 \leq v \leq m$. The optimization is then cycled over all views while keeping the previously obtained $\mathbf{U}^{(\cdot)}$'s fixed.

### 2.3 Centroid-Based Co-regularization

In this section, we present an alternative regularization scheme that regularizes each view-specific set of eigenvectors $\mathbf{U}^{(v)}$ towards a common centroid $\mathbf{U}^*$ (akin to a *consensus* set of eigenvectors) . In contrast with the pairwise regularization approach which has $\binom{m}{2}$ pairwise regularization terms, where $m$ is the number of views, the centroid based regularization scheme has $m$ pairwise regularization terms. The objective function can be written as:

$$\max_{\mathbf{U}^{(1)},\mathbf{U}^{(2)},...,\mathbf{U}^{(m)},\mathbf{U}^*\in\mathbb{R}^{n \times k}} \sum_{v=1}^{m} tr \left( \mathbf{U}^{(v)^T} \mathcal{L}^{(v)} \mathbf{U}^{(v)} \right) + \sum_{v} \lambda_v tr \left( \mathbf{U}^{(v)} \mathbf{U}^{(v)^T} \mathbf{U}^* \mathbf{U}^{*^T} \right),$$

$$\text{s.t.} \quad \mathbf{U}^{(v)^T} \mathbf{U}^{(v)} = I, \ \forall \ 1 \leq v \leq V, \quad \mathbf{U}^{*^T} \mathbf{U}^* = I \quad (7)$$

This objective tries to balance a trade-off between the individual spectral clustering objectives and the agreement of each of the view-specific eigenvectors $\mathbf{U}^{(v)}$ with the consensus eigenvectors $\mathbf{U}^*$. Each regularization term is weighted by a parameter $\lambda_v$ specific to that view, where $\lambda_v$ can be set to reflect the importance of view $v$.

Just like for Equation 6, the objective in Equation 7 can be solved in an alternating fashion optimizing each of the $\mathbf{U}^{(v)}$'s one at a time, keeping all other variables fixed, followed by optimizing the consensus $\mathbf{U}^*$, keeping all the $\mathbf{U}^{(v)}$'s fixed.

It is easy to see that with all other view-specific eigenvectors and the consensus $\mathbf{U}^*$ fixed, optimizing $\mathbf{U}^{(v)}$ for view $v$ amounts to solving the following:

$$\max_{\mathbf{U}^{(v)}\in\mathbb{R}^{n \times k}} tr \left( \mathbf{U}^{(v)^T} \mathcal{L}^{(v)} \mathbf{U}^{(v)} \right) + \lambda_v tr \left( \mathbf{U}^{(v)} \mathbf{U}^{(v)^T} \mathbf{U}^* \mathbf{U}^{*^T} \right), \quad \text{s.t.} \quad \mathbf{U}^{(v)^T} \mathbf{U}^{(v)} = I \quad (8)$$

which is nothing but equivalent to solving the standard spectral clustering objective for $\mathbf{U}^{(v)}$ with a modified Laplacian $\mathcal{L}^{(v)} + \lambda_v \mathbf{U}^* \mathbf{U}^{*^T}$. Solving for the consensus $\mathbf{U}^*$ requires solving the following objective:

$$\max_{\mathbf{U}^*\in\mathbb{R}^{n \times k}} \sum_{v} \lambda_v tr \left( \mathbf{U}^{(v)} \mathbf{U}^{(v)^T} \mathbf{U}^* \mathbf{U}^{*^T} \right), \quad \text{s.t.} \quad \mathbf{U}^{*^T} \mathbf{U}^* = I \quad (9)$$

Using the circular property of matrix traces, Equation 9 can be rewritten as:

$$\max_{\mathbf{U}^*\in\mathbb{R}^{n \times k}} tr \left\{ \mathbf{U}^{*^T} \left( \sum_{v} \lambda_v \left( \mathbf{U}^{(v)} \mathbf{U}^{(v)^T} \right) \right) \mathbf{U}^* \right\}, \quad \text{s.t.} \quad \mathbf{U}^{*^T} \mathbf{U}^* = I \quad (10)$$

which is equivalent to solving the standard spectral clustering objective for $\mathbf{U}^*$ with a modified Laplacian $\sum_{v} \lambda_v \left( \mathbf{U}^{(v)} \mathbf{U}^{(v)^T} \right)$. In contrast with the pairwise co-regularization approach of Section 2.1 which computes optimal view specific eigenvectors $\mathbf{U}^{(v)}$'s, which finally need to be combined (e.g., via column-wise concatenation) before running the $k$-means step, the centroid-based co-regularization approach directly finds an optimal $\mathbf{U}^*$ to be used in the $k$-means step. One possible downside of the centroid-based co-regularization approach is that noisy views could potentially affect the optimal $\mathbf{U}^*$ as it depends on all the views. To deal with this, careful selection of the weighing parameter $\lambda_v$ is required. If it is *a priori* known that some views are noisy, then it is advisable to use a small value of $\lambda_v$ for such views, so as to prevent them from adversely affecting $\mathbf{U}^*$.

## 3 Experiments

We compare both of our co-regularization based multi-view spectral clustering approaches with a number of baselines. In particular, we compare with:

- **Single View:** Using the most informative view, i.e., one that achieves the best spectral clustering performance using a single view of the data.
- **Feature Concatenation:** Concatenating the features of each view, and then running standard spectral clustering using the graph Laplacian derived from the joint view representation of the data.
- **Kernel Addition:** Combining different kernels by adding them, and then running standard spectral clustering on the corresponding Laplacian. As suggested in earlier findings [9], even this seemingly simple approach often leads to near optimal results as compared to more sophisticated approaches for classification. It can be noted that kernel addition reduces to feature concatenation for the special case of linear kernel. In general, kernel addition is same as concatenation of features in the Reproducing Kernel Hilbert Space.
- **Kernel Product (element-wise):** Multiplying the corresponding entries of kernels and applying standard spectral clustering on the resultant Laplacian. For the special case of Gaussian kernel, element-wise kernel product would be same as simple feature concatenation if both kernels use same width parameter $\sigma$. However, in our experiments, we use different width parameters for different views so the performances of kernel product may not be directly comparable to feature concatenation.
- **CCA based Feature Extraction:** Applying CCA for feature fusion from multiple views of the data [10], and then running spectral clustering using these extracted features. We apply both standard CCA and kernel CCA for feature extraction and report the clustering results for whichever gives the best performance.
- **Minimizing-Disagreement Spectral Clustering:** Our last baseline is the *minimizing-disagreement* approach to spectral clustering [11], and is perhaps most closely related to our co-regularization based approach to spectral clustering. This algorithm is discussed more in Sec. 4.

To distinguish between the results of our two co-regularization based approaches, in the tables containing the results, we use symbol "P" to denote the *pairwise* co-regularization method and symbol "C" to denote the *centroid* based co-regularization method. For datasets with more than 2 views, we have also explicitly mentioned the number of views in parentheses.

We report experimental results on two synthetic and three real-world datasets. We give a brief description of each dataset here.

- **Synthetic data 1:** Our first synthetic dataset consists of two views and is generated in a manner akin to [12] which first chooses the cluster $c_i$ each sample belongs to, and then generates each of the views $x_i^{(1)}$ and $x_i^{(2)}$ from a two-component Gaussian mixture model. These views are combined to form the sample $(x_i^{(1)}, x_i^{(2)}, c_i)$. We sample 1000 points from each view. The cluster means in view 1 are $\mu_1^{(1)} = (1\ 1)$, $\mu_2^{(1)} = (2\ 2)$, and in view 2 are $\mu_1^{(2)} = (2\ 2)$, $\mu_2^{(2)} = (1\ 1)$. The covariances for the two views are given below.

$$\Sigma_1^{(1)} = \begin{pmatrix} 1 & 0.5 \\ 0.5 & 1.5 \end{pmatrix}, \Sigma_1^{(2)} = \begin{pmatrix} 0.3 & 0 \\ 0 & 0.6 \end{pmatrix}, \Sigma_2^{(1)} = \begin{pmatrix} 0.3 & 0 \\ 0 & 0.6 \end{pmatrix}, \Sigma_2^{(2)} = \begin{pmatrix} 1 & 0.5 \\ 0.5 & 1.5 \end{pmatrix}$$

- **Synthetic data 2:** Our second synthetic dataset consists of three views. Moreover, the features are correlated. Each view still has two clusters. Each view is generated by a two component Gaussian mixture model. The cluster means in view 1 are $\mu_1^{(1)} = (1\ 1)$, $\mu_2^{(1)} = (3\ 4)$; in view 2 are $\mu_1^{(2)} = (1\ 2)$, $\mu_2^{(2)} = (2\ 2)$; and in view 3 are $\mu_1^{(3)} = (1\ 1)$, $\mu_2^{(3)} = (3\ 3)$. The covariances for the three views are given below. The notation $\Sigma_c^{(v)}$ denotes the parameter for $c$'th cluster in $v$'th view.

$$\Sigma_1^{(1)} = \begin{pmatrix} 1 & 0.5 \\ 0.5 & 1.5 \end{pmatrix}, \quad \Sigma_1^{(2)} = \begin{pmatrix} 1 & -0.2 \\ -0.2 & 1 \end{pmatrix}, \quad \Sigma_1^{(3)} = \begin{pmatrix} 1.2 & 0.2 \\ 0.2 & 1 \end{pmatrix}$$

$$\Sigma_2^{(1)} = \begin{pmatrix} 0.3 & 0.2 \\ 0.2 & 0.6 \end{pmatrix}, \quad \Sigma_2^{(2)} = \begin{pmatrix} 0.6 & 0.1 \\ 0.1 & 0.5 \end{pmatrix}, \quad \Sigma_2^{(3)} = \begin{pmatrix} 1 & 0.4 \\ 0.4 & 0.7 \end{pmatrix}$$

- **Reuters Multilingual data:** The test collection contains feature characteristics of documents originally written in five different languages (English, French, German, Spanish and Italian), and their translations, over a common set of 6 categories [13]. This corpus is built by sampling parts of the Reuters RCV1 and RCV2 collections [14, 15]. We use documents originally in English as the first view and their French translations as the second view. We randomly sample 1200 documents from this collection in a balanced manner, with each of the 6 clusters having 200 documents. The documents are in bag-of-words representation which implies that the features are extremely sparse and high-dimensional. The standard similarity measures (like Gaussian kernel) in very high dimensions are often unreliable. Since spectral clustering essentially works with similarities of the data, we first project the data using Latent Semantic Analysis (LSA) [16] to a 100-dimensional space and compute similarities in this lower dimensional space. This is akin to a computing topic based similarity of documents [17].
- **UCI Handwritten digits data:** Our second real-world dataset is taken from the handwritten digits (0-9) data from the UCI repository. The dataset consists of 2000 examples, with view-1 being the 76 Fourier coefficients, and view-2 being the 216 profile correlations of each example image.
- **Caltech-101 data:** Our third real-world dataset is a subset of the Caltech-101 data from the Multiple Kernel Learning repository from which we chose 450 examples having 30 underlying clusters. We experiment with 4 kernels from this dataset. In particular, we chose the "pixel features", the "Pyramid Histogram Of Gradients", bio-inspired "Sparse Localized Features", and SIFT descriptors as our four views. We report results on our co-regularized spectral clustering for two, three and four views cases.

We use normalized mutual information (NMI) as the clustering quality evaluation measure, which gives the mutual information between obtained clustering and the true clustering normalized by the cluster entropies. NMI ranges between 0 and 1 with higher value indicating closer match to the true clustering. We use Gaussian kernel for computing the graph similarities in all the experiments, unless mentioned otherwise. The standard deviation of the kernel is taken equal to the median of the pair-wise Euclidean distances between the data points. In our experiments, the co-regularization parameter $\lambda$ is varied from 0.01 to 0.05 and the best result is reported (we keep $\lambda$ the same for all views; one can however also choose different $\lambda$'s based on the importance of individual views). We experiment with $\lambda$ values more exhaustively later in this Section where we show that our approach outperforms other baselines for a wide range of $\lambda$. In the results table, the numbers in the parentheses are the standard deviations of the performance measures obtained with 20 different runs of $k$-means with random initializations.

### 3.1 Results
The results for all datasets are shown in Table 1. For two-view synthetic data (Synthetic Data 1), both the co-regularized spectral clustering approaches outperform all the baselines by a significant margin, with the pairwise approach doing marginally better than the centroid-based approach. The closest performing approaches are kernel addition and CCA. For synthetic data, order-2 polynomial kernel based kernel-CCA gives best performance among all CCA variants, while Gaussian kernel based kernel-CCA performs poorly. We do not report results for Gaussian kernel CCA here. All the multi-view baselines outperform the single view case for the synthetic data.

For three-view synthetic data (Synthetic Data 2), we can see that simple feature concatenation does not help much. In fact, it reduces the performance when the third view is added, so we report the performance with only two views for feature concatenation. Kernel addition with three views gives a good improvement over single view case. As compared to other baselines (with two views), both our co-regularized spectral clustering approaches with two views perform better. For both approaches, addition of third view also results in improving the performance beyond the two view case.

For the document clustering results on Reuters multilingual data, English and French languages are used as the two views. On this dataset too, both our approaches outperform all the baselines by a significant margin. The next best performance is attained by minimum-disagreement spectral clustering [11] approach. It should be noted that CCA and element-wise kernel product performances are worse than that of single view.

For UCI Handwritten digits dataset, quite a few approaches including kernel addition, element-wise kernel multiplication, and minimum-disagreement are close to both of our co-regularized spectral

| Method | Synth data 1 | Synth data 2 | Reuters | Handwritten | Caltech |
|---|---|---|---|---|---|
| Best Single View | 0.267 (0.0) | 0.898 (0.0) | 0.287 (0.019) | 0.641 (0.008) | 0.510 (0.008) |
| Feature Concat | 0.294 (0.0) | 0.923 (0.0) | 0.298 (0.020) | 0.619 (0.015) | – |
| Kernel Addition | 0.339 (0.0) | 0.973 (0.0) | 0.323 (0.021) | 0.744 (0.030) | 0.383 (0.008) |
| Kernel Product | 0.277 (0.0) | 0.959 (0.0) | 0.123 (0.010) | 0.754 (0.026) | 0.429 (0.007) |
| CCA | 0.330 (0.0) | 0.932 (0.0) | 0.147 (0.003) | 0.682 (0.019) | 0.466 (0.007) |
| Min-Disagreement | 0.313 (0.0) | 0.936 (0.0) | 0.342 (0.024) | 0.745 (0.024) | 0.389 (0.008) |
| Co-regularized (P) (2) | 0.378 (0.0) | 0.981 (0.0) | 0.375 (0.002) | 0.759 (0.031) | 0.527 (0.007) |
| Co-regularized (P) (3) | – | 0.989 (0.0) | – | – | 0.533 (0.008) |
| Co-regularized (P) (4) | – | – | – | – | 0.564 (0.007) |
| Co-regularized (C) (2) | 0.367 (0.0) | 0.955 (0.0) | 0.360 (0.025) | 0.768 (0.025) | 0.522 (0.004) |
| Co-regularized (C) (3) | – | 0.989 (0.0) | – | – | 0.512 (0.007) |
| Co-regularized (C) (4) | – | – | – | – | 0.561 (0.005) |

Table 1: NMI results on various datasets for different baselines and the proposed approaches. Numbers in parentheses are the std. deviations. The numbers (2), (3) and (4) indicate the number of views used in our co-regularized spectral clustering approach. Other multi-view baselines were run with maximum number of views available (or maximum number of views they can handle). Letters (P) and (C) indicate pairwise and centroid based regularizations respectively.

clustering approaches. It can be also be noted that feature concatenation actually performs worse than single view on this dataset.

For Caltech-101 data, we cannot do feature concatenation since only kernels are available. Surprisingly, on this dataset, all the baselines perform worse than the single view case. On the other hand, both of our co-regularized spectral clustering approaches with two views outperform the single view case. As we added more views that were available for the Caltech-101 datasets, we found that the performance of the pairwise approach consistently went up as we added the third and the fourth view. On the other hand, the performance of the centroid-based approach slightly got worse upon adding the third view (possibly due to the view being noisy which affected the learned $\mathbf{U}^*$); however addition of the fourth view brought the performance almost close to that of the pairwise case.

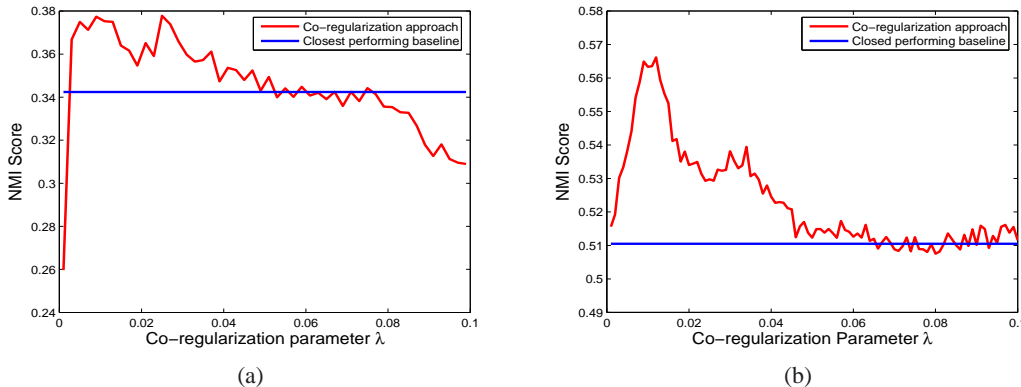

(a)                                            (b)

Figure 1: NMI scores of Co-regularized Spectral Clustering as a function of $\lambda$ for (a) **Reuters multilingual data** and (b) **Caltech-101 data**

We also experiment with various values of co-regularization parameter $\lambda$ and observe its effect on the clustering performance. Our reported results are for the pairwise co-regularization approach. Similar trends were observed for the centroid-based co-regularization approach and therefore we do not report them here. Fig. 1(a) shows the plot for Reuters multilingual data. The NMI score shoots up right after $\lambda$ starts increasing from 0 and reaches a peak at $\lambda = 0.01$. After reaching a second peak at about 0.025, it starts decreasing and hovers around the second best baseline (Minimizing-disagreement in this case) for a while. The NMI becomes worse than the second best baseline after $\lambda = 0.075$. The plot for Caltech-101 data is shown in Fig. 1(b). The normalized mutual information (NMI) starts increasing as the value of lambda is increased away from 0, and reaches a peak at $\lambda = 0.01$. It starts to decrease after that with local ups and downs. For the range of $\lambda$ shown in the plot, the NMI for co-regularized spectral clustering is greater than the closest baseline for most of

the $\lambda$ values. These results indicate that although the performance of our algorithms depends on the weighing parameter $\lambda$, it is reasonably stable across a wide range of $\lambda$.

## 4 Related Work

A number of clustering algorithms have been proposed in the past to learn with multiple views of the data. Some of them first extract a set of shared features from the multiple views and then apply any off-the-shelf clustering algorithm such as $k$-means on these features. The Canonical Correlation Analysis (CCA) [2, 10] based approach is an example of this. Alternatively, some other approaches exploit the multiple views of the data as part of the clustering algorithm itself. For example, [19] proposed an Co-EM based framework for multi-view clustering in mixture models. Co-EM approach computes expected values of hidden variables in one view and uses these in the M-step for other view, and vice versa. This process is repeated until a suitable stopping criteria is met. The algorithm often does not converge.

Multi-view clustering algorithms have also been proposed in the framework of spectral clustering [11, 20, 21]. In [20], the authors obtain a graph cut which is good on average over the multiple graphs but may not be the best for a single graph. They give a random walk based formulation for the problem. [11] approaches the problem of two-view clustering by constructing a bipartite graph from nodes of both views. Edges of the bipartite graph connect nodes from one view to those in the other view. Subsequently, they solve standard spectral clustering problem on this bipartite graph. In [21], a co-training based framework is proposed where the similarity matrix of one view is constrained by the eigenvectors of the Laplacian in the other view. In [22], the information from multiple graphs are fused using Linked Matrix Factorization. Consensus clustering approaches can also be applied to the problem of multi-view clustering [7]. These approaches do not generally work with original features. Instead, they take different clusterings of a dataset coming from different sources as input and reconcile them to find a final clustering.

## 5 Discussion

We proposed a multi-view clustering approach in the framework of spectral clustering. The approach uses the philosophy of co-regularization to make the clusterings in different views agree with each other. Co-regularization idea has been used in the past for semi-supervised learning problems. To the best of our knowledge, this is the first work to apply the idea to the problem of unsupervised learning, in particular to spectral clustering. The co-regularized spectral clustering has a joint optimization function for spectral embeddings of all the views. An alternating maximization framework reduces the problem to the standard spectral clustering objective which is efficiently solvable using state-of-the-art eigensolvers.

It is possible to extend the proposed framework to the case where some of the views have missing data. For missing data points, the corresponding entries in the similarity matrices would be unavailable. We can estimate these missing similarities by the corresponding similarities in other views. One possible approach to estimate the missing entry could be to simply average the similarities from views in which the data point is available. Proper normalization of similarities (possibly by Frobenius norm of the whole matrix) might be needed before averaging to make them comparable. Other methods for missing kernel entries estimation can also be used. It is also possible to assign weights to different views in the proposed objective function as done in [20], if we have some a priori knowledge about the informativeness of the views.

Our co-regularization based framework can also be applied to other unsupervised problems such as spectral methods for dimensionality reduction. For example, the Kernel PCA algorithm [23] can be extended to work with multiple views by defining each view as having its own Kernel PCA objective function and having a regularizer which enforces the embeddings to look *similar* across all views (e.g., by enforcing the similarity matrices defined on embeddings of each view to be close to each other). Theoretical analysis of the proposed approach can also be pursued as a separate line of work. There has been very little prior work analyzing spectral clustering methods. For instance, there has been some work on consistency analysis of single view spectral clustering [24], which provides results about the rate of convergence as the sample size increases, using tools from theory of linear operators and empirical processes. Similar convergence properties could be studied for multi-view spectral clustering. We can expect the convergence to be faster for multi-view case. Co-regularization reduces the size of hypothesis space and hence less number of examples should be needed to converge to a solution.

# References

[1] A. Blum and T. Mitchell. Combining labeled and unlabeled data with co-training. In *Conference on Learning Theory*, 1998.

[2] Kamalika Chaudhuri, Sham M. Kakade, Karen Livescu, and Karthik Sridharan. Multi-view Clustering via Canonical Correlation Analysis. In *International Conference on Machine Learning*, 2009.

[3] Ulrike von Luxburg. A Tutorial on Spectral Clustering. *Statistics and Computing*, 2007.

[4] J. Shi and J. Malik. Normalized cuts and Image Segmentation. *IEEE Transactions on Pattern Analysis and Machine Intelligence*, 22:888–905, 1997.

[5] A. Ng, M. Jordan, and Y. Weiss. On spectral clustering: analysis and an algorithm. In *Advances in Neural Information Processing Systems*, 2002.

[6] Vikas Sindhwani, Partha Niyogi, and Mikhail Belkin. A Co-regularization approach to semi-supervised learning with multiple views. In *Proceedings of the Workshop on Learning with Multiple Views, International Conference on Machine Learning*, 2005.

[7] Alexander Strehl and Joydeep Ghosh. Cluster Ensembles - A Knowledge Reuse Framework for Combining Multiple Partitions. *Journal of Machine Learning Research*, pages 583–617, 2002.

[8] Donglin Niu, Jennifer G. Dy, and Michael I. Jordan. Multiple non-redundant spectral clustering views. In *International Conference on Machine Learning*, 2010.

[9] Corinna Cortes, Mehryar Mohri, and Afshin Rostamizadeh. Learning non-linear combination of kernels. In *Advances in Neural Information Processing Systems*, 2009.

[10] Matthew B. Blaschko and Christoph H. Lampert. Correlational Spectral Clustering. In *Computer Vision and Pattern Recognition*, 2008.

[11] Virginia R. de Sa. Spectral Clustering with two views. In *Proceedings of the Workshop on Learning with Multiple Views, International Conference on Machine Learning*, 2005.

[12] Xing Yi, Yunpeng Xu, and Changshui Zhang. Multi-view em algorithm for finite mixture models. In *ICAPR, Lecture Notes in Computer Science, Springer-Verlag*, 2005.

[13] Massih-Reza Amini, Nicolas Usunier, and Cyril Goutte. Learning from multiple partially observed views - an application to multilingual text categorization. In *Advances in Neural Information Processing Systems*, 2009.

[14] D. D. Lewis, Y. Yang, T. Rose, and F. Li. RCV1. A new benchmark collection for text categorization research. *Journal of Machine Learning Research*, 5:361–397, 2004.

[15] Reuters. Corpus, volume 2, multilingual corpus, 1996-08-20 to 1997-08-19, 2005.

[16] Thomas Hofmann. Probabilistic latent semantic analysis. In *Uncertainty in Artificial Intelligence*, pages 289–296, 1999.

[17] David M. Blei, Andreq Y. Ng, and Michael I. Jordan. Latent Dirichlet Allocation. *Journal of Machine Learning Research*, pages 993–1022, 2003.

[18] The UCSD Multiple Kernel Learning Repository. http://mkl.ucsd.edu.

[19] Steffen Bickel and Tobias Scheffer. Multi-View Clustering. In *IEEE International Conference on Data Mining*, 2004.

[20] Dengyong Zhou and Christopher J. C. Burges. Spectral Clustering and Transductive Learning with Multiple Views. In *International Conference on Machine Learning*, 2007.

[21] Abhishek Kumar and Hal Daumé. A Co-training Approach for Multiview Spectral Clustering. In *International Conference on Machine Learning*, 2011.

[22] Wei Tang, Zhengdong Lu, and Inderjit S. Dhillon. Clustering with Multiple Graphs. In *IEEE International Conference on Data Mining*, 2009.

[23] Y. Bengio, P. Vincent, and J.F. Paiement. Spectral clustering and kernel PCA are learning eigenfunctions. *Technical Report 2003s-19, CIRANO*, 2003.

[24] Ulrike von Luxburg, Mikhail Belkin, and Olivier Bousquet. Consistency of Spectral Clustering. *Annals of Statistics*, 36(2):555–586, 2008.

